# Adaptive Regularization for Transductive Support Vector Machine

**Zenglin Xu** [†‡]
[†] Cluster MMCI
Saarland Univ. & MPI INF
Saarbrucken, Germany
zlxu@mpi-inf.mpg.de

**Rong Jin**
Computer Sci. & Eng.
Michigan State Univ.
East Lansing, MI, U.S.
rongjin@cse.msu.edu

**Jianke Zhu**
Computer Vision Lab
ETH Zurich
Zurich, Switzerland
zhuji@vision.ee.ethz.ch

**Irwin King**[‡]    **Michael R. Lyu**[‡]
[‡] Computer Science & Engineering
The Chinese Univ. of Hong Kong
Shatin, N.T., Hong Kong
{king,lyu}@cse.cuhk.edu.hk

**Zhirong Yang**
Information & Computer Science
Helsinki Univ. of Technology
Espoo, Finland
zhirong.yang@tkk.fi

## Abstract

We discuss the framework of Transductive Support Vector Machine (TSVM) from the perspective of the regularization strength induced by the unlabeled data. In this framework, SVM and TSVM can be regarded as a learning machine without regularization and one with full regularization from the unlabeled data, respectively. Therefore, to supplement this framework of the regularization strength, it is necessary to introduce data-dependant partial regularization. To this end, we reformulate TSVM into a form with controllable regularization strength, which includes SVM and TSVM as special cases. Furthermore, we introduce a method of adaptive regularization that is data dependant and is based on the smoothness assumption. Experiments on a set of benchmark data sets indicate the promising results of the proposed work compared with state-of-the-art TSVM algorithms.

## 1 Introduction

Semi-supervised learning has attracted a lot of research focus in recently years. Most of the existing approaches can be roughly divided into two categories: (1) the clustering-based methods [12, 4, 8, 17] assume that most of the data, including both the labeled ones and the unlabeled ones, should be far away from the decision boundary of the target classes; (2) the manifold-based methods make the assumption that most of data lie on a low-dimensional manifold in the input space, which include Label Propagation [21], Graph Cuts [2], Spectral Kernels [9, 22], Spectral Graph Transducer [11], and Manifold Regularization [1]. The comprehensive study on semi-supervised learning techniques can be found in the recent surveys [23, 3].

Although semi-supervised learning wins success in many real-world applications, there still remains two major unsolved challenges. One is whether the unlabeled data can help the classification, and the other is what is the relation between the clustering assumption and the manifold assumption.

As for the first challenge, Singh et al. [16] provided a finite sample analysis on the usefulness of unlabeled data based on the cluster assumption. They show that unlabeled data may

be useful for improving the error bounds of supervised learning methods when the margin between different classes satisfies some conditions. However, in the real-world problems, it is hard to identify the conditions that unlabeled data can help.

On the other hand, it is interesting to explore the relation between the low density assumption and the manifold assumption. Narayanan et al. [14] implied that the cut-size of the graph partition converges to the weighted volume of the boundary which separates the two regions of the domain for a fixed partition. This makes a step forward for exploring the connection between graph-based partitioning and the idea surrounding the low density assumption. Unfortunately, this approach cannot be generalized uniformly over all partitions. Lafferty and Wasserman [13] revisited the assumptions of semi-supervised learning from the perspective of minimax theory, and suggested that the manifold assumption is stronger than the smoothness assumption for regression. Till now, the underlying relationships between the cluster assumption and the manifold assumption are still undisclosed. Specifically, it is unclear that in what kind of situation the clustering assumption or the manifold assumption should be adopted.

In this paper, we address these current limitations by a unified solution from the perspective of the regularization strength of the unlabeled data. Taking Transductive Support Vector Machine (TSVM) as an example, we suggest an framework that introduces the regularization strength of the unlabeled data when estimating the decision boundary. Therefore, we can obtain a spectrum of models by varying the regularization strength of unlabeled data which corresponds to changing the models from supervised SVM to Transductive SVM. To select the optimal model under the proposed framework, we employ the manifold regularization assumption that enables the prediction function to be smooth over the data space. Further, the optimal function is a linear combination of supervised models, weakly semi-supervised models, and semi-supervised models. Additionally, it provides an effective approach towards combining the cluster assumption and the manifold assumption in semi-supervised learning.

The rest of this paper is organized as follows. In Section 2, we review the background of Transductive SVM. In Section 3, we first present a framework of models with different regularization strength, followed by an integrating approach based on manifold regularization. In Section 4, we report the experimental results on a series of benchmark data sets. Section 5 concludes the paper.

## 2    Related Work on TSVM

Before presenting the formulation of TSVM, we first describe the notations used in this paper. Let $\mathcal{X} = (\mathbf{x}_1, \ldots, \mathbf{x}_n)$ denote the entire data set, including both the labeled examples and the unlabeled ones. We assume that the first $l$ examples within $\mathcal{X}$ are labeled and the next $n - l$ examples are unlabeled. We denote the unknown labels by $\mathbf{y}_u = (y_{l+1}^u, \ldots, y_n^u)$.

TSVM [12] maximizes the margin in the presence of unlabeled data and keeps the boundary traversing through low density regions while respecting labels in the input space. Under the maximum-margin framework, TSVM aims to find the classification model with the maximum classification margin for both labeled and unlabeled examples, which amounts to solve the following optimization problem:

$$\min_{\mathbf{w} \in \mathbb{R}^n, \mathbf{y}_u \in \mathbb{R}^{n-\ell}, \xi \in \mathbb{R}^n} \quad \frac{1}{2} \|\mathbf{w}\|_{\mathbf{K}} + C \sum_{i=1}^{l} \xi_i + C^* \sum_{i=l+1}^{n} \xi_i \tag{1}$$

$$\text{s. t.} \quad y_i \mathbf{w}^\top \phi(\mathbf{x}_i) \geq 1 - \xi_i, \ \xi_i \geq 0, \ 1 \leq i \leq l,$$
$$y_i^u \mathbf{w}^\top \phi(\mathbf{x}_i) \geq 1 - \xi_i, \ \xi_i \geq 0, \ l + 1 \leq i \leq n,$$

where $C$ and $C^*$ are the trade-off parameters between the complexity of the function $\mathbf{w}$ and the margin errors. Moreover, the prediction function can be formulated as $f(\mathbf{x}) = \mathbf{w}^\top \phi(\mathbf{x})$. Note that we remove the bias term in the above formulation, since it can be taken into account by introducing a constant element into the input pattern alternatively.

As in [19] and [20], we can rewrite (1) into the following optimization problem:

$$\min_{\mathbf{f},\xi} \quad \frac{1}{2}\mathbf{f}^\top \mathbf{K}^{-1}\mathbf{f} + C\sum_{i=1}^{l}\xi_i + C^*\sum_{i=l+1}^{n}\xi_i \tag{2}$$

$$\text{s. t.} \quad y_i f_i \geq 1 - \xi_i, \ \xi_i \geq 0, \ 1 \leq i \leq l,$$
$$|f_i| \geq 1 - \xi_i, \ \xi_i \geq 0, \ l+1 \leq i \leq n.$$

The optimization problem held in TSVM is a non-linear non-convex optimization [6]. During past several years, researchers have devoted a significant amount of research efforts to solving this critical problem. A branch-and-bound method [5] was developed to search for the optimal solution, which is only limited to solve the problem with a small number of examples due to involving the heavy computational cost. To apply TSVM for large-scale problems, Joachims [12] proposed a label-switching-retraining procedure to speed up the optimization procedure. Later, the hinge loss in TSVM is replaced by a smooth loss function, and a gradient descent method is used to find the decision boundary in a region of low density [4]. In addition, there are some iterative methods, such as deterministic annealing [15], concave-convex procedure (CCCP) [8], and convex relaxation method [19, 18]. Despite the success of TSVM, the unlabeled data not necessarily improve classification accuracy.

To better utilize the unlabeled data, unlike existing TSVM approaches, we propose a framework that tries to control the regularization strength of the unlabeled data. To do this, we intend to learn the optimal regularization strength configuration from the combination of a spectrum of models: supervised, weakly-supervised, and semi-supervised.

## 3 TSVM: A Regularization View

For the sake of illustration, we first study a model that does not penalize on the classification errors of unlabeled data. Note that the penalization on the margin errors of unlabeled data can be included if needed. Therefore, we have the following form of TSVM that can be derived through the duality:

$$\min_{\mathbf{f},\xi} \quad \frac{1}{2}\mathbf{f}^\top \mathbf{K}^{-1}\mathbf{f} + C\sum_{i=1}^{l}\xi_i \tag{3}$$

$$\text{s. t.} \quad y_i f_i \geq 1 - \xi_i, \ \xi_i \geq 0, \ 1 \leq i \leq l,$$
$$f_i^2 \geq 1, \ l+1 \leq i \leq n.$$

### 3.1 Full Regularization of Unlabeled Data

In order to adjust the strength of the regularization raised from the unlabeled examples, we introduce a coefficient $\rho \geq 0$, and modify the above problem (3) as below:

$$\min_{\mathbf{f},\xi} \quad \frac{1}{2}\mathbf{f}^\top \mathbf{K}^{-1}\mathbf{f} + C\sum_{i=1}^{l}\xi_i \tag{4}$$

$$\text{s. t.} \quad y_i f_i \geq 1 - \xi_i, \ \xi_i \geq 0, \ 1 \leq i \leq l,$$
$$f_i^2 \geq \rho, \ l+1 \leq i \leq n.$$

Obviously, it is the standard TSVM for $\rho = 1$. In particular, the larger the $\rho$ is, the stronger the regularization of unlabeled data is. It is also important to note that we only take into account the classification errors on the labeled examples in the above equation. Namely, we only denote $\xi_i$ for each labeled example.

Further, we write $\mathbf{f} = (\mathbf{f}_l; \mathbf{f}_u)$ where $\mathbf{f}_l = (f_1, \ldots, f_l)$ and $\mathbf{f}_u = (f_{l+1}, \ldots, f_n)$ represent the prediction for the labeled and the unlabeled examples, respectively. According to the inverse lemma of the block matrix, we can write $\mathbf{K}^{-1}$ as follows:

$$\mathbf{K}^{-1} = \begin{pmatrix} \mathbf{M}_l^{-1} & -\mathbf{K}_{l,l}^{-1}\mathbf{K}_{l,u}\mathbf{M}_u^{-1} \\ -\mathbf{M}_u^{-1}\mathbf{K}_{u,l}\mathbf{K}_{l,l}^{-1} & \mathbf{M}_u^{-1} \end{pmatrix},$$

where

$$\mathbf{M}_l = \mathbf{K}_{l,l} - \mathbf{K}_{l,u}\mathbf{K}_{u,u}^{-1}\mathbf{K}_{u,l},$$

$$\mathbf{M}_u = \mathbf{K}_{u,u} - \mathbf{K}_{u,l}\mathbf{K}_{l,l}^{-1}\mathbf{K}_{l,u}.$$

Thus, the term $\mathbf{f}^\top \mathbf{K}^{-1}\mathbf{f}$ is computed as

$$\mathbf{f}^\top \mathbf{K}^{-1}\mathbf{f} \;\;=\;\; \mathbf{f}_l^\top \mathbf{M}_l^{-1}\mathbf{f}_l + \mathbf{f}_u^\top \mathbf{M}_u^{-1}\mathbf{f}_u - 2\mathbf{f}_l^\top \mathbf{K}_{l,l}^{-1}\mathbf{K}_{l,u}\mathbf{M}_u^{-1}\mathbf{f}_u.$$

When the unlabeled data are loosely correlated to the labeled data, namely when most of the elements within $\mathbf{K}_{u,l}$ are small, this leads to $\mathbf{M}_u \approx \mathbf{K}_u$. We refer to this case as "*weakly unsupervised learning*". Using the above equations, we rewrite TSVM as follows:

$$\min_{\mathbf{f}_l,\mathbf{f}_u,\xi} \quad \frac{1}{2}\mathbf{f}_l^\top \mathbf{M}_l^{-1}\mathbf{f}_l + C\sum_{i=1}^{l}\xi_i + \omega(\mathbf{f}_l,\rho) \tag{5}$$
$$\text{s. t.} \quad y_i f_i \geq 1 - \xi_i, \; \xi_i \geq 0, \; 1 \leq i \leq l,$$

where $\omega(\mathbf{f}_l,\rho)$ is a regularization function for $\mathbf{f}_l$ and it is the result of the following optimization problem:

$$\min_{\mathbf{f}_u} \quad \frac{1}{2}\mathbf{f}_u^\top \mathbf{M}_u^{-1}\mathbf{f}_u - \mathbf{f}_l^\top \mathbf{K}_{l,l}^{-1}\mathbf{K}_{l,u}\mathbf{M}_u^{-1}\mathbf{f}_u \tag{6}$$
$$\text{s. t.} \quad [f_i^u]^2 \geq \rho, \quad l+1 \leq i \leq n.$$

To understand the regularization function $\omega(\mathbf{f}_l,\rho)$, we first compute the dual of the problem (6) by the Lagrangian function:

$$\mathcal{L} \;\;=\;\; \frac{1}{2}\mathbf{f}_u^\top \mathbf{M}_u^{-1}\mathbf{f}_u - \mathbf{f}_l^\top \mathbf{K}_{l,l}^{-1}\mathbf{K}_{l,u}\mathbf{M}_u^{-1}\mathbf{f}_u - \sum_{i=1}^{n_u}\frac{1}{2}\lambda_i([f_i^u]^2 - \rho)$$

$$\;\;=\;\; \frac{1}{2}\mathbf{f}_u^\top (\mathbf{M}_u^{-1} - \mathrm{D}(\lambda))\mathbf{f}_u - \mathbf{f}_l^\top \mathbf{K}_{l,l}^{-1}\mathbf{K}_{l,u}\mathbf{M}_u^{-1}\mathbf{f}_u + \frac{\rho}{2}\lambda^\top \mathbf{e},$$

where $\mathrm{D}(\lambda) = \mathrm{diag}(\lambda_1,\ldots,\lambda_{n-l})$ and $\mathbf{e}$ denotes a vector with all elements being one. As the derivatives vanish for optimality, we have

$$\mathbf{f}_u \;\;=\;\; (\mathbf{M}_u^{-1} - \mathrm{D}(\lambda))^{-1}\mathbf{M}_u^{-1}\mathbf{K}_{u,l}\mathbf{K}_{l,l}^{-1}\mathbf{f}_l$$

$$\;\;=\;\; (\mathbf{I} - \mathbf{M}_u \mathrm{D}(\lambda))^{-1}\mathbf{K}_{u,l}\mathbf{K}_{l,l}^{-1}\mathbf{f}_l,$$

where $\mathbf{I}$ is an identity matrix.

Replacing $\mathbf{f}_u$ in (6) with the above equation, we have the following dual problem:

$$\max_{\lambda} \quad -\frac{1}{2}\mathbf{f}_l^\top \mathbf{K}_{l,l}^{-1}\mathbf{K}_{l,u}(\mathbf{M}_u - \mathbf{M}_u \mathrm{D}(\lambda)\mathbf{M}_u)^{-1}\mathbf{K}_{u,l}\mathbf{K}_{l,l}^{-1}\mathbf{f}_l + \rho\lambda^\top \mathbf{e} \tag{7}$$
$$\text{s. t.} \quad \mathbf{M}_u^{-1} \succeq \mathrm{D}(\lambda), \; \lambda_i \geq 0, \; i = 1,\ldots,n-l.$$

The above formulation allows us to understand how the parameter $\rho$ controls the strength of regularization from the unlabeled data. In the following, we will show that a series of learning models can be derived through assigning various values for the coefficient $\rho$.

## 3.2 No Regularization from Unlabeled Data

First, we study the case of $\rho = 0$. We have the following theorem to illustrate the relationship between the dual problem (7) and the supervised SVM.

**Theorem 1** *When $\rho = 0$, the optimization problem is reduced to the standard supervised SVM.*

**Proof 1** *It is not difficult to see that the optimal solution to (7) is $\lambda = \mathbf{0}$. As a result, $\omega(\mathbf{f}_l, \rho)$ becomes*

$$\omega(\mathbf{f}_l, \rho = 0) \quad = \quad -\frac{1}{2}\mathbf{f}_l \mathbf{K}_{l,l}^{-1} \mathbf{K}_{l,u} \mathbf{M}_u^{-1} \mathbf{K}_{u,l} \mathbf{K}_{l,l}^{-1} \mathbf{f}_l$$

*Substituting $\omega(\mathbf{f}_l, \rho)$ in (5) with the formulation above, the overall optimization problem becomes*

$$\min_{\mathbf{f}_l, \xi} \quad \frac{1}{2}\mathbf{f}_l^\top (\mathbf{M}_l^{-1} - \mathbf{K}_{l,l}^{-1} \mathbf{K}_{l,u} \mathbf{M}_u^{-1} \mathbf{K}_{u,l} \mathbf{K}_{l,l}^{-1})\mathbf{f}_l + C\sum_{i=1}^{l} \xi_i$$

$$s.\ t. \quad y_i f_i \geq 1 - \xi_i,\ \xi_i \geq 0,\ 1 \leq i \leq l.$$

*According to the matrix inverse lemma, we calculate $\mathbf{M}_l^{-1}$ as below:*

$$
\begin{aligned}
\mathbf{M}_l^{-1} \quad &= \quad (\mathbf{K}_{l,l} - \mathbf{K}_{l,u} \mathbf{K}_{u,u}^{-1} \mathbf{K}_{u,l})^{-1} \\
&= \quad \mathbf{K}_{l,l}^{-1} + \mathbf{K}_{l,l}^{-1} \mathbf{K}_{l,u} (\mathbf{K}_{u,u} - \mathbf{K}_{u,l} \mathbf{K}_{l,l}^{-1} \mathbf{K}_{l,u})^{-1} \mathbf{K}_{u,l} \mathbf{K}_{l,l}^{-1} \\
&= \quad \mathbf{K}_{l,l}^{-1} + \mathbf{K}_{l,l}^{-1} \mathbf{K}_{l,u} \mathbf{M}_u^{-1} \mathbf{K}_{u,l} \mathbf{K}_{l,l}^{-1}.
\end{aligned}
$$

*Finally, the optimization problem is simplified as*

$$\min_{\mathbf{f}_l, \xi} \quad \frac{1}{2}\mathbf{f}_l^\top \mathbf{K}_{l,l}^{-1} \mathbf{f}_l + C\sum_{i=1}^{l} \xi_i \qquad (8)$$

$$s.\ t. \quad y_i f_i \geq 1 - \xi_i,\ \xi_i \geq 0,\ 1 \leq i \leq l.$$

*Clearly, the above optimization is identical to the standard supervised SVM. Hence, the unlabeled data are not employed to regularize the decision boundary when $\rho = 0$.*

### 3.3 Partial Regularization of Unlabeled Data

Second, we consider the case when $\rho$ is small. According to (7), we expect $\lambda$ to be small when $\rho$ is small. As a result, we can approximate $(\mathbf{M}_u - \mathbf{M}_u D(\lambda)\mathbf{M}_u)^{-1}$ as follows:

$$(\mathbf{M}_u - \mathbf{M}_u D(\lambda)\mathbf{M}_u)^{-1} \approx \mathbf{M}_u^{-1} + D(\lambda).$$

Consequently, we can write $\omega(\mathbf{f}_l, \rho)$ as follows:

$$\omega(\mathbf{f}_l, \rho) \quad = \quad -\frac{1}{2}\mathbf{f}_l^\top \mathbf{K}_{l,l}^{-1} \mathbf{K}_{l,u} \mathbf{M}_u^{-1} \mathbf{K}_{u,l} \mathbf{K}_{l,l}^{-1} \mathbf{f}_l + \phi(\mathbf{f}_l, \rho), \qquad (9)$$

where $\phi(\mathbf{f}_l, \rho)$ is the output of the following optimization problem

$$\max_{\lambda} \quad \rho\lambda^\top \mathbf{e} - \frac{1}{2}\mathbf{f}_l^\top \mathbf{K}_{l,l}^{-1} \mathbf{K}_{l,u} D(\lambda) \mathbf{K}_{u,l} \mathbf{K}_{l,l}^{-1} \mathbf{f}_l$$

$$s.\ t. \quad \mathbf{M}_u^{-1} \succeq D(\lambda),\ \lambda_i \geq 0,\ i = 1, \ldots, n - l.$$

We can simplify the above problem by approximating $\mathbf{M}_u^{-1} \succeq D(\lambda)$ as $\lambda_i \leq [\sigma_1(\mathbf{M}_u)]^{-1}$, $i = 1, \ldots, n - l$, where $\sigma_1(\mathbf{M}_u)$ represents the maximum eigenvalue of matrix $\mathbf{M}_u$. The resulting simplified problem becomes

$$\max_{\lambda} \quad \frac{\rho}{2}\lambda^\top \mathbf{e} - \frac{1}{2}\mathbf{f}_l^\top \mathbf{K}_{l,l}^{-1} \mathbf{K}_{l,u} D(\lambda) \mathbf{K}_{u,l} \mathbf{K}_{l,l}^{-1} \mathbf{f}_l$$

$$s.\ t. \quad 0 \leq \lambda_i \leq [\sigma_1(\mathbf{M}_u)]^{-1},\ 1 \leq i \leq n - l.$$

As the above problem is a linear programming problem, the solution for $\lambda$ can be computed as:

$$\lambda_i = \begin{cases} 0 & [\mathbf{K}_{u,l} \mathbf{K}_{l,l}^{-1} \mathbf{f}_l]_i^2 > \rho, \\ \sigma(\mathbf{M}_u)^{-1} & [\mathbf{K}_{u,l} \mathbf{K}_{l,l}^{-1} \mathbf{f}_l]_i^2 \leq \rho. \end{cases}$$

From the above formulation, we find that $\rho$ plays the role of a threshold of selecting the unlabeled examples. Since $[\mathbf{K}_{u,l} \mathbf{K}_{l,l}^{-1} \mathbf{f}_l]_i$ can be regarded as the approximation for the $i$th

unlabeled example, the above formulation can be interpreted in the way that only the unlabeled examples with low prediction confidence will be selected for regularizing the decision boundary. Moreover, all the unlabeled examples with high prediction confidence will be ignored. From the above discussions, we can conclude that $\rho$ determines the regularization strength of unlabeled examples.

Then, we rewrite the overall optimization problem as below:

$$\min_{\mathbf{f}_l,\xi} \max_{\lambda} \quad \frac{1}{2}\mathbf{f}_l^\top \mathbf{K}_{l,l}^{-1}\mathbf{f}_l + C\sum_{i=1}^{l}\xi_i - \frac{1}{2}\mathbf{f}_l^\top \mathbf{K}_{l,l}^{-1}\mathbf{K}_{l,u}\mathrm{D}(\lambda)\mathbf{K}_{u,l}\mathbf{K}_{l,l}^{-1}\mathbf{f}_l \qquad (10)$$

$$\text{s. t.} \quad y_i f_i \geq 1 - \xi_i,\ \xi_i \geq 0,\ 1 \leq i \leq l,$$
$$0 \leq \lambda_i \leq [\sigma_1(\mathbf{M}_u)]^{-1},\ 1 \leq i \leq n - l.$$

This is a min-max optimization problem and thus the global optimal solution can be guaranteed. To obtain the optimal solution, we employ an alternating optimization procedure, which iteratively computes the values of $\mathbf{f}_l$ and $\lambda$. To account for the penalty on the margin error from the unlabeled data, we just need to add an extra constraint of $\lambda_i \leq 2C$ for $i = 1, \ldots, n - l$.

By varying the parameter $\rho$ from 0 to 1, we can indeed obtain a series of transductive models for SVM. When $\rho$ is small, we call the corresponding optimization problem as weakly semi-supervised learning. Therefore, it is important to find an appropriate $\rho$ which adapts for the input data. However, as the data distribution is usually unknown, it is very challenging to directly estimate an optimal regularization strength parameter $\rho$. Instead, we try to explore an alternative approach to select an appropriate $\rho$ by combining the prediction functions. Due to the large cost in calculating the inverse of kernel matrices, one can solve the dual problems according to the Representer theorem.

### 3.4 Adaptive Regularization

As stated in previous sections, $\rho$ determines the regularization strength of the unlabeled data. We now try to adapt the parameter $\rho$ according to the unlabeled data information. Specifically, we intend to implicitly select the best $\rho$ from a given list, i.e., $\Upsilon = \{\rho_1, \ldots, \rho_m\}$ where $\rho_1 = 0$ and $\rho_m = 1$. This is equivalent to selecting the optimal $\mathbf{f}$ from a list of prediction functions, i.e., $\mathcal{F} = \{\mathbf{f}_1, \ldots, \mathbf{f}_m\}$. Motivated from the ensemble technique for semi-supervised learning [7], we assume that the optimal $\mathbf{f}$ comes from a linear combination of the base functions $\{\mathbf{f}_i\}$. We then have:

$$f = \sum_{i=1}^{m}\theta_i f_i,\ \sum_{i=1}^{m}\theta_i = 1,\ \theta_i \geq 0,\ i = 1, \ldots, m.$$

where $\theta_i$ is the weight of the prediction function $f_i$ and $\theta \in \mathrm{R}^m$. One can also involve a priori to $\theta_i$. For example, if we have more confidences on the semi-supervised classifier, we can introduce a constraint like $\theta_m \geq 0.5$. It is important to note that the learning functions in ensemble methods [7] are usually weak learners, while in our approach, the learning functions are strong learners with different degrees of regularization.

In the following, we study how to set the regularization strength adaptive to data. Since TSVM naturally follows the cluster assumption of semi-supervised learning, in order to complement the cluster assumption, we adopt another principle in semi-supervised learning, i.e., the manifold assumption. From the point of view of manifold assumption in semi-supervised learning, the prediction function $f$ should be smooth on unlabeled data. To this end, the approach of manifold regularization is widely adopted as a smoothing term in semi-supervised learning literatures, e.g., [1, 10]. In the following, we will employ the manifold regularization principle for selecting the regularization strength.

The manifold regularization is mainly based on a graph $\mathcal{G} = < \mathcal{V}, \mathcal{E} >$ derived from the whole data space $\mathbf{X}$, where $\mathcal{V} = \{\mathbf{x}_i\}_{i=1}^{n}$ is the vertex set, and $\mathcal{E}$ denotes the edges linking pairs of nodes. In general, a graph is built in the following four steps: (1) constructing adjacency graph; (2) calculating the weights on edges; (3) computing the adjacency matrix $\mathbf{W}$; (4)

obtaining the graph Laplacian by $\mathcal{L} = \mathrm{diag}(\sum_{j=1}^{n} W_{ij}) - \mathbf{W}$. Then, we denote the manifold regularization term as $f^{\top}\mathcal{L}f$.

For simplicity, we denote the predicted values of function $f_i$ on the data $\mathbf{X}$ as $\mathbf{f}_i$, such that $\mathbf{f}_i = ([f_i]_1, \ldots, [f_i]_n)$. $\mathbf{F} = (\mathbf{f}_1, \ldots, \mathbf{f}_m)^{\top}$ is used to represent the set of the prediction values of all prediction functions. Finally, We have the following minimization problem:

$$\min_{\theta} \quad \frac{1}{2}\eta(\theta^{\top}\mathbf{F})\mathcal{L}(\mathbf{F}^{\top}\theta) - \mathbf{y}_{\ell}^{\top}(\mathbf{F}_{\ell}^{\top}\theta) \tag{11}$$
$$\text{s. t.} \quad \theta^{\top}\mathbf{e} = 1, \ \theta_i \geq 0, \ i = 1, \ldots, m,$$

where the second term, $\mathbf{y}_{\ell}^{\top}(\mathbf{F}_{\ell}^{\top}\theta)$, is used to strengthen the confidence on the prediction over the labeled data. $\eta$ is a trade-off parameter. The above optimization problem is a simple quadratic programming problem, which can be solved very efficiently. It is important to note that the above optimization problem is less sensitive to the graph structure than Laplacian SVM as used in [1], since the basic learning functions are all strong learners. It also saves a huge amount of efforts in estimating the parameters compared with Laplacian SVM.

The above approach indeed provides a practical approach towards a combination of both the cluster assumption and the manifold assumption. It is empirically suggested that combining these two assumptions helps to improve the prediction accuracy of semi-supervised learning according to the survey paper on semi-supervised SVM [6]. Moreover, when $\rho = 0$, supervised models are incorporated in the framework. Thus the usefulness of unlabeled in naturally considered by the regularization. This therefore provides a practical solution to the problems described in Section 1.

## 4   Experiment

In this section, we give details of our implementation and discuss the results on several benchmark data sets for our proposed approach. To conduct a comprehensive evaluation, we employ several well-known datasets as the testbed. As summarized in Table 1, three image data sets and five text data sets are selected from the recent book (`www.kyb.tuebingen.mpg.de/ssl-book/`) and the literature (`www.cs.uchicago.edu/~vikass/`).

Table 1: Datasets used in our experiments. $d$ represents the data dimensionality, and $n$ denotes the total number of examples.

| Data set | $n$ | $d$ | Data set | $n$ | $d$ |
|---|---|---|---|---|---|
| usps | 1500 | 241 | digit1 | 1500 | 241 |
| coil | 1500 | 241 | ibm vs rest | 1500 | 11960 |
| pcmac | 1946 | 7511 | page | 1051 | 3000 |
| link | 1051 | 1800 | pagelink | 1051 | 4800 |

For simplicity, our proposed adaptive regularization approach is denoted as ARTSVM. To evaluate it, we conduct an extensive comparison with several state-of-the-art approaches, including the label-switching-retraining algorithm in SVM-Light [12], CCCP [8], and $\nabla$TSVM [4]. We employ SVM as the baseline method.

In our experiments, we repeat all the algorithms 20 times for each dataset. In each run, 10% of the data are randomly selected as the training data and the remaining data are used as the unlabeled data. The value of C in all algorithms are selected from $[1, 10, 100, 1000]$ using cross-validation. The set of $\rho$ is set to $[0, 0.01, 0.05, 0.1, 1]$ and $\eta$ is fixed to 0.001. As stated in Section 3.4, ARTSVM is less sensitive to the graph structure. Thus, we adopt a simple way to construct the graph: for each data, the number of neighbors is set to 20 and binary weighting is employed. In ARTSVM, the supervised, weakly semi-supervised, and semi-supervised algorithms are based on implementation in LibSVM (`www.csie.ntu.edu.tw/~cjlin/libsvm/`), MOSEK (`www.mosek.org`), and $\nabla$TSVM (`www.kyb.tuebingen.mpg.de/bs/people/chapelle/lds/`), respectively. For the comparison algorithms, we adopt the original authors' own implementations.

Table 2 summarizes the classification accuracy and the standard deviations of the proposed ARTSVM method and other competing methods. We can draw several observations from

the results. First of all, we can clearly see that our proposed algorithm performs significantly better than the baseline SVM method across all the data sets. Note that some very large deviations in SVM are mainly because the labeled data and the unlabeled data may have quite different distributions after the random sampling. On the other hand, the unlabeled data capture the underlying distribution and help to correct such random error. Comparing ARTSVM with other TSVM algorithms, we observe that ARTSVM achieves the best performance in most cases. For example, for the digital image data sets, especially digit1, supervised learning usually works well and the advantages of TSVM are very limited. However, the proposed ARTSVM outperforms both the supervised and other semi-supervised algorithms. This indicates that the appropriate regularization from the unlabel data improves the classification performance.

Table 2: The classification performance of Transductive SVMs on benchmark data sets.

| Data Set | ARTSVM | $\nabla$TSVM | SVM | CCCP | SVM-light |
|---|---|---|---|---|---|
| usps | **81.30**$\pm$4.04 | 79.44$\pm$3.63 | 79.23$\pm$8.60 | 80.48$\pm$3.20 | 78.16$\pm$4.41 |
| digit1 | **82.10**$\pm$2.11 | 80.55$\pm$1.94 | 81.70$\pm$5.61 | 80.69$\pm$2.97 | 77.53$\pm$4.24 |
| coil | **81.70**$\pm$2.10 | 79.84$\pm$1.88 | 78.98$\pm$8.07 | 80.15$\pm$2.90 | 79.03$\pm$2.84 |
| ibm vs rest | **78.04**$\pm$1.44 | 76.83$\pm$2.11 | 72.90$\pm$2.32 | 77.52$\pm$1.51 | 73.99$\pm$5.18 |
| pcmac | **95.50**$\pm$0.88 | 95.42$\pm$0.95 | 92.57$\pm$0.82 | 94.86$\pm$1.09 | 91.42$\pm$7.24 |
| page | 94.65$\pm$1.19 | **94.78**$\pm$1.83 | 75.22$\pm$17.38 | 94.47$\pm$1.67 | 93.98$\pm$2.60 |
| link | **94.27**$\pm$0.97 | 93.56$\pm$1.58 | 40.79$\pm$3.63 | 92.60$\pm$2.10 | 92.18$\pm$2.45 |
| pagelink | **97.31**$\pm$0.68 | 96.53$\pm$1.84 | 89.41$\pm$3.12 | 95.97$\pm$2.22 | 94.89$\pm$1.81 |

## 5    Conclusion

This paper presents a novel framework for semi-supervised learning from the perspective of the regularization strength from the unlabeled data. In particular, for Transductive SVM, we show that SVM and TSVM can be incorporated as special cases within this framework. In more detail, the loss on the unlabeled data can essentially be regarded as an additional regularizer for the decision boundary in TSVM. To control the regularization strength, we introduce an alternative method of data-dependant regularization based on the principle of manifold regularization. Empirical studies on benchmark data sets demonstrate that the proposed framework is more effective than the previous transductive algorithms and purely supervised methods.

For future work, we plan to design a controlling strategy that is adaptive to data from the perspective of low density assumption and manifold regularization of semi-supervised learning. Finally, it is desirable to integrate the low density assumption and manifold regularization into a unified framework.

## Acknowledgement

The work was supported by the National Science Foundation (IIS-0643494), National Institute of Health (1R01GM079688-01), Research Grants Council of Hong Kong (CUHK4158/08E and CUHK4128/08E), and MSRA (FY09-RES-OPP-103). It is also affiliated with the MS-CUHK Joint Lab for Human-centric Computing & Interface Technologies.

## References

[1] Mikhail Belkin, Partha Niyogi, and Vikas Sindhwani. Manifold regularization: A geometric framework for learning from labeled and unlabeled examples. *Journal of Machine Learning Research*, 7:2399–2434, 2006.

[2] Avrim Blum and Shuchi Chawla. Learning from labeled and unlabeled data using graph mincuts. In *ICML '01: Proceedings of the 18th international conference on Machine learning*, pages 19–26. Morgan Kaufmann, San Francisco, CA, 2001.

[3] O. Chapelle, B. Schölkopf, and A. Zien, editors. *Semi-Supervised Learning*. MIT Press, Cambridge, MA, 2006.

[4] O. Chapelle and A. Zien. Semi-supervised classification by low density separation. In *Proceedings of the Tenth International Workshop on Artificial Intelligence and Statistics*, pages 57–64, 2005.

[5] Olivier Chapelle, Vikas Sindhwani, and Sathiya Keerthi. Branch and bound for semi-supervised support vector machines. In B. Schölkopf, J. Platt, and T. Hoffman, editors, *Advances in Neural Information Processing Systems 19*. MIT Press, Cambridge, MA, 2007.

[6] Olivier Chapelle, Vikas Sindhwani, and Sathiya S. Keerthi. Optimization techniques for semi-supervised support vector machines. *Journal of Machine Learning Research*, 9:203–233, 2008.

[7] Ke Chen and Shihai Wang. Regularized boost for semi-supervised learning. In J.C. Platt, D. Koller, Y. Singer, and S. Roweis, editors, *Advances in Neural Information Processing Systems 20*, pages 281–288. MIT Press, Cambridge, MA, 2008.

[8] Ronan Collobert, Fabian Sinz, Jason Weston, and Léon Bottou. Large scale transductive SVMs. *Journal of Machine Learning Researech*, 7:1687–1712, 2006.

[9] S. C. H. Hoi, M. R. Lyu, and E. Y. Chang. Learning the unified kernel machines for classification. In *Proceedings of Twentith International Conference on Knowledge Discovery and Data Mining (KDD-2006)*, pages 187–196, New York, NY, USA, 2006. ACM Press.

[10] Steven C. H. Hoi, Rong Jin, and Michael R. Lyu. Learning nonparametric kernel matrices from pairwise constraints. In *ICML '07: Proceedings of the 24th international conference on Machine learning*, pages 361–368, New York, NY, USA, 2007. ACM.

[11] T. Joachims. Transductive learning via spectral graph partitioning. In *ICML '03: Proceedings of the 20th international conference on Machine learning*, pages 290–297, 2003.

[12] Thorsten Joachims. Transductive inference for text classification using support vector machines. In *ICML '99: Proceedings of the 16th international conference on Machine learning*, pages 200–209, San Francisco, CA, USA, 1999. Morgan Kaufmann Publishers Inc.

[13] John Lafferty and Larry Wasserman. Statistical analysis of semi-supervised regression. In J.C. Platt, D. Koller, Y. Singer, and S. Roweis, editors, *Advances in Neural Information Processing Systems 20*, pages 801–808. MIT Press, Cambridge, MA, 2008.

[14] Hariharan Narayanan, Mikhail Belkin, and Partha Niyogi. On the relation between low density separation, spectral clustering and graph cuts. In B. Schölkopf, J. Platt, and T. Hoffman, editors, *Advances in Neural Information Processing Systems 19*, pages 1025–1032. MIT Press, Cambridge, MA, 2007.

[15] Vikas Sindhwani, S. Sathiya Keerthi, and Olivier Chapelle. Deterministic annealing for semi-supervised kernel machines. In *ICML '06: Proceedings of the 23rd international conference on Machine learning*, pages 841–848, New York, NY, USA, 2006. ACM Press.

[16] Aarti Singh, Robert Nowak, and Xiaojin Zhu. Unlabeled data: Now it helps, now it doesn't. In D. Koller, D. Schuurmans, Y. Bengio, and L. Bottou, editors, *Advances in Neural Information Processing Systems 21*, pages 1513–1520. 2009.

[17] Junhui Wang, Xiaotong Shen, and Wei Pan. On efficient large margin semisupervised learning: Method and theory. *Journal of Machine Learning Research*, 10:719–742, 2009.

[18] Linli Xu and Dale Schuurmans. Unsupervised and semi-supervised multi-class support vector machines. In *AAAI*, pages 904–910, 2005.

[19] Zenglin Xu, Rong Jin, Jianke Zhu, Irwin King, and Michael R. Lyu. Efficient convex relaxation for transductive support vector machine. In J.C. Platt, D. Koller, Y. Singer, and S. Roweis, editors, *Advances in Neural Information Processing Systems 20*, pages 1641–1648. MIT Press, Cambridge, MA, 2008.

[20] T. Zhang and R. Ando. Analysis of spectral kernel design based semi-supervised learning. In Y. Weiss, B. Schölkopf, and J. Platt, editors, *Advances in Neural Information Processing Systems (NIPS 18)*, pages 1601–1608. MIT Press, Cambridge, MA, 2006.

[21] Dengyong Zhou, Olivier Bousquet, Thomas Navin Lal, Jason Weston, and Bernhard Schölkopf. Learning with local and global consistency. In Sebastian Thrun, Lawrence Saul, and Bernhard Schölkopf, editors, *Advances in Neural Information Processing Systems 16*. MIT Press, Cambridge, MA, 2004.

[22] X. Zhu, J. Kandola, Z. Ghahramani, and J. Lafferty. Nonparametric transforms of graph kernels for semi-supervised learning. In *Advances in Neural Information Processing Systems (NIPS 17)*, pages 1641–1648, Cambridge, MA, 2005. MIT Press.

[23] Xiaojin Zhu. Semi-supervised learning literature survey. Technical report, Computer Sciences, University of Wisconsin-Madison, 2005.
